# Simulation of the Neocognitron on a CCD Parallel Processing Architecture

**Michael L. Chuang and Alice M. Chiang**
M.I.T Lincoln Laboratory
Lexington, MA 02173
e-mail: *chuang@micro.ll.mit.edu*

## Abstract

The neocognitron is a neural network for pattern recognition and feature extraction. An analog CCD parallel processing architecture developed at Lincoln Laboratory is particularly well suited to the computational requirements of shared-weight networks such as the neocognitron, and implementation of the neocognitron using the CCD architecture was simulated. A modification to the neocognitron training procedure, which improves network performance under the limited arithmetic precision that would be imposed by the CCD architecture, is presented.

## 1 INTRODUCTION

Multilayer neural networks characterized by local interlayer connectivity and groups of nodes that are constrained to have the same weights on their input lines are often refered to as shared-weight networks. A group of nodes with identical weights where each node is connected to a different portion of the layer immediately beneath can be thought of as a collection of spatially replicated receptive fields. Among the desirable attributes of shared-weight networks is the fact that substantially less storage is required for weights than would be required by a more conventional network with a comparable number of nodes. Furthermore, reducing the number of free parameters through use of shared weights and local receptive fields, as opposed to simply reducing the number of hidden nodes, may be an effective way of obtaining good generalization when only a small training set is available (Martin and Pittman, 1989). However, the most immediately obvious attribute of a shared-weight architecture is that the replicated receptive fields allow a learned feature to be detected anywhere within the input. This feature is particularly useful in

tasks where position invariance is required (Le Cun, 1989). Neural networks using shared weights have been applied successfully to areas ranging from handwritten digit recognition (Le Cun, Boser, et. al., 1989) to phoneme extraction in speech preprocessing (Waibel, et. al., 1989).

A CCD architecture that is well suited to implementing shared-weight networks has been developed at Lincoln Laboratory (Chiang and LaFranchise, 1991). This architecture performs high-speed inner product computations and is able to accommodate the often complicated data access patterns of a shared-weight network without imposing the burden of this complexity on the host computer; input and output to devices built using this architecture are simple. The neocognitron (Fukushima, 1988) was selected as a candidate for implementation by the CCD architecture. In particular, we were interested the effect that limited precision arithmetic might have on network performance.

## 2   THE NEOCOGNITRON

The neocognitron is a multilayer feed-forward neural network for pattern recognition. The nodes or cells in each layer or level of the neocognitron are further subdivided into cell planes, where all the nodes in a given cell plane are feature detectors tuned to the same feature but connected to a different portion of the level immediately beneath (the first level has cell planes connected directly to the input). Each cell plane can be viewed as an array of identical, overlapping receptive fields.

Three types of processing elements or nodes are used in the neocognitron. S-cells perform feature extraction, c-cells compensate for local shifts of features, and v-cells are intended to prevent random excitation of s-cells. A given cell plane contains only one type of node. A cell plane containing only s-cells, for example, is thus called an s-plane. Each level of the network contains several s-planes, an identical number of c-planes, and exactly one v-plane. The function of an s-cell is to generate a nonlinear function of the inner product of a stored weight template $a_\lambda(k, \kappa, i, j)$ and the contents of its receptive field. (In this notation $\lambda$ denotes the level of the s-plane with which the template is associated, and the $k$ and $\kappa$ indicate the particular s- and c-planes between which the template serves as a connection. The $i, j$ are spatial coordinates within the template.) An s-plane is therefore a feature map of its input. Each c-plane is paired with a single s-plane of the same level. A c-cell has a small receptive field on its corresponding s-plane and performs a weighted average of the values of the s-cells to which it is connected. This implements a form of local feature-shift invariance, and a c-plane is a feature map of its input which is unchanged by small translations of features in the input. A schematic of a three-level neocognitron is shown in Figure 1.

The cell planes in the first level of the network typically correspond to maps of simple features such as oriented line segments. The second level of the neocognitron is given the output of the first-level c-planes as input, and tends to form more complicated features from the first-level cell planes. Successively higher levels correspond to even more complex features; at the top level, each c-cell (of which there is exactly one in each top-level c-plane) corresponds to one input pattern in a trained neocognitron. The basic idea is to break up each input pattern into simple components such as line segments and corners, then to put the pieces back together, allowing a certain

An image feature extractor (IFE) device suitable for performing the inner products required by a neural network with local receptive fields and shared weights has been fabricated (Chiang and LaFranchise, 1991). The IFE consists of a 775-stage CCD tapped delay line for holding and shifting input pixels or node values, 49 eight-bit, four-quadrant multiplying digital-to-analog converters (MDACs), and on-chip storage for 980 eight-bit digital weights. Figure 2 is a photomicrograph of the chip, which has an area of 29 mm$^2$ and performs over one billion arithmetic operations/second when clocked at 10 MHz. The device dissipates less than 1 W.

The 49 MDACs of the IFE are arranged in a $7 \times 7$ array; each MDAC nondestructively senses the value held in an appropriate point along the 775-stage tapped delay line, which holds six 128-pixel lines, plus seven pixels of the following line, of the input image. Image pixels are continuously loaded into the device in row-by-row fashion. Each MDAC has a local memory of twenty eight-bit digital weights for holding inner product kernel or template values. Conceptually, the device scans a $7 \times 7$ "window" over an input array, shifting one position at each step, and computes the inner product of each of the twenty templates with the portion of the image beneath the window. The multiplications of each inner product are performed in parallel and the partial sums are connected to a common output line, allowing the complete inner product to be computed in one clock. In actuality, the device passes the input image under the $7 \times 7$ window, performing twenty inner products with each shift of the image. A schematic of data flow through the IFE device is shown in Figure 3.

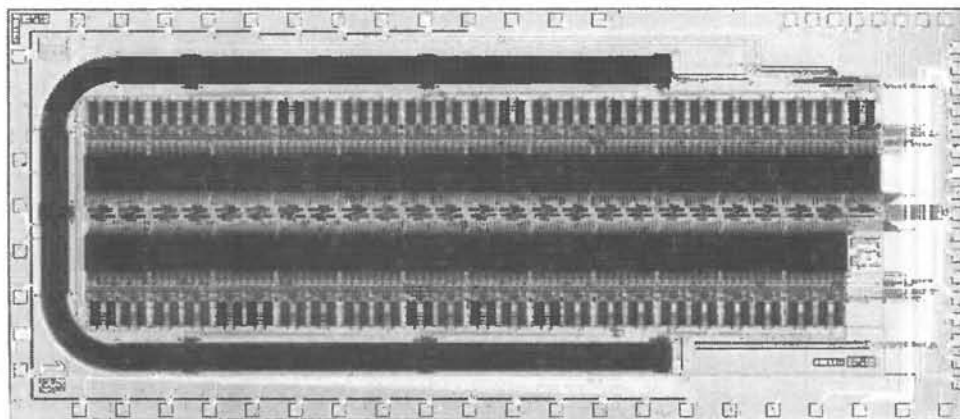

Figure 2: Photomicrograph of the CCD Image Feature Extractor

# 4   A MODIFIED TRAINING ALGORITHM

Most computer simulations of the neocognitron have used floating point arithmetic as well as weights which are, for all practical purposes, real numbers. However, a neocognitron implemented using an IFE device would use fairly low precision

amount of relative position shift between the pieces at each stage of reassembly. This allows the network to identify deformed or shifted inputs. The extent to which a particular network is able to tolerate deformation of input patterns depends on the amount of overlap between adjacent receptive fields as well as the size and weighting of c-cell receptive fields.

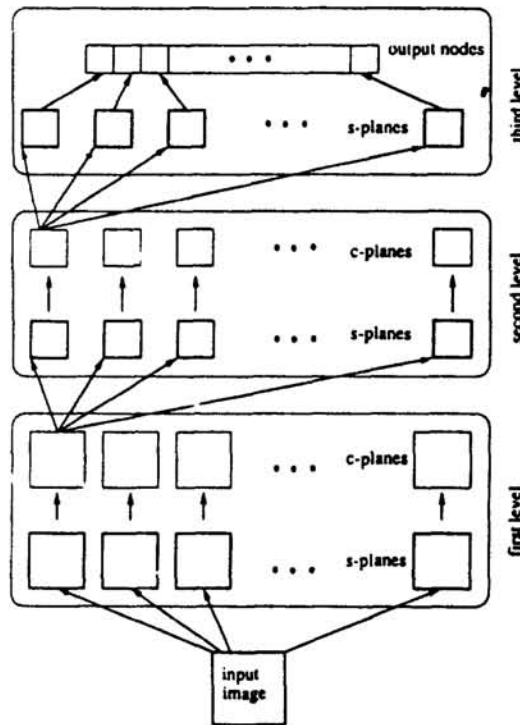

The output of an s-cell is given by

$$s_\lambda(k,m,n) = \begin{cases} 0, & z < 0 \\ r_\lambda z, & z \geq 0 \end{cases}$$

where

$$z = \frac{\sigma_\lambda + \sum_{\kappa=1}^{K_{\lambda-1}} \sum_{i=1}^{I} \sum_{j=1}^{J} a_\lambda(k,\kappa,i,j) \cdot c_{\lambda-1}(\kappa, m+i-1, n+j-1)}{\sigma_\lambda + \frac{r_\lambda}{1+r_\lambda} \cdot b_\lambda(k) \cdot v_\lambda(m,n)} - 1$$

and c-cells compute

$$c_\lambda(k,m,n) = \begin{cases} 0, & y < 0 \\ \dfrac{y}{1+y}, & y \geq 0 \end{cases}$$

where

$$y = \sum_{i=1}^{I} \sum_{j=1}^{J} d_\lambda(i,j) \cdot s_\lambda(k, m+i-1, n+j-1)$$

Figure 1: Schematic of a Three-Level Neocognitron

The majority of the computation in the neocognitron consists of the inner products. A good implementation of shared-weight networks such as the neocognitron must be capable of performing high speed inner product computations as well as supporting the data access patterns of the algorithm efficiently. A device which meets both these requirements is described in the following section.

## 3　THE IMAGE FEATURE EXTRACTOR

The neocognitron is most easily visualized as a three-dimensional structure built of the s-, c- and v-cells, but the s- and c-planes can be generated by raster scanning weight templates whose values are the $a_\lambda(k, \kappa, i, j)$ or the $d_\lambda(i, j)$, respectively, over the appropriate input. This operation can be performed efficiently by the CCD architecture alluded to in the Introduction. In this architecture, analog node values are represented using charge packets while fully programmable weight values are stored digitally on-chip. The multiplications of the generic weighted sum computation are performed in parallel, with the summation performed in the charge domain, yielding a complete inner product sum on each clock.

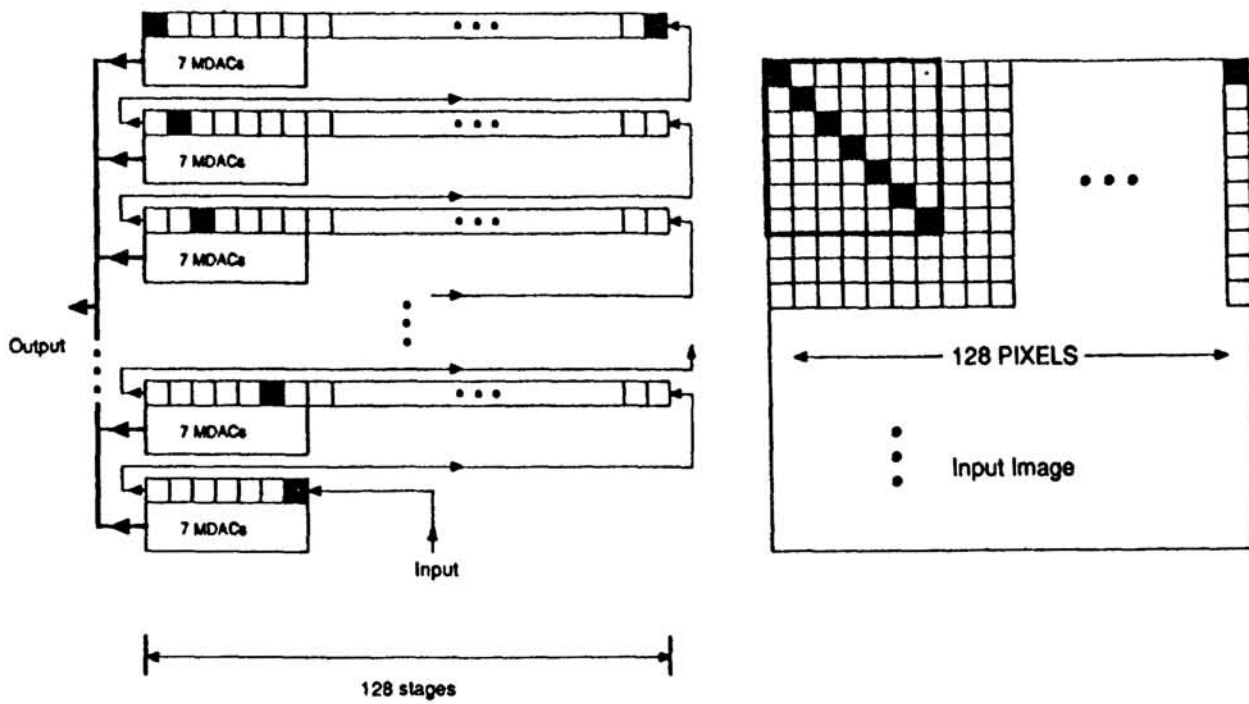

Figure 3: Dataflow in the Image Feature Extractor

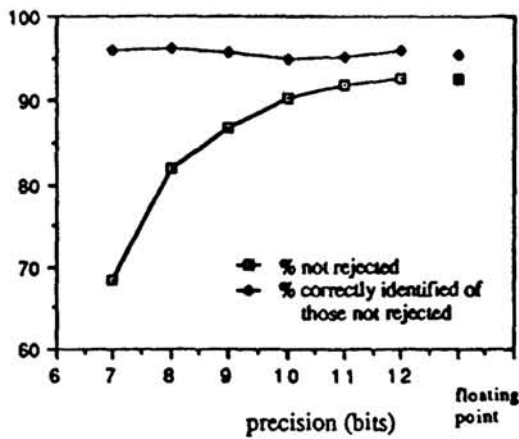

(a)

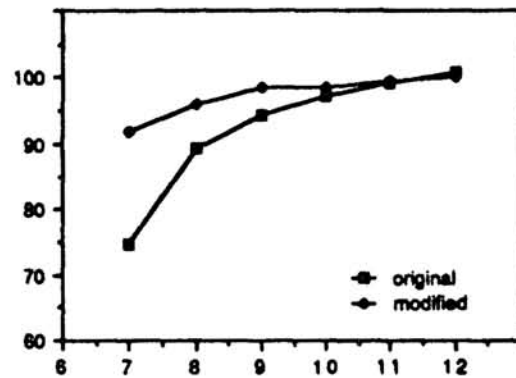

(b)

Figure 4: (a) Effect of Arithmetic Precision on Classification (b) Comparision of Original and Modified Training Procedures

arithmetic and quantized weights. In order to determine whether the neocognitron
would continue to perform under such restrictions, a software simulation of neocog-
nitrons using low precision arithmetic was implemented. Weights were taken from
a network that was previously trained using floating point arithmetic and quantized
to a number of bits equal to the arithmetic precision. As can be seen from Figure
4(a), the fraction of inputs correctly identified (bottom curve) from a test set of
handwritten letters decreases substantially as arithmetic precision is reduced. Al-
though the error rate (top curve) remains approximately constant, lower arithmetic
precision tends to increase the number of rejections.

## 4.1   EFFECT OF LIMITED PRECISION

Inspection of the weights revealed that the range of weights from previously trained
nets was too large to be represented using the number of bits available. Either small
weights were set to zero, large weights were clipped, or both. Networks trained using
low precision arithmetic tended to group all input patterns into a single category.
This can again be attributed to the restricted range of possible weight values. The
neocognitron training algorithm consists of assigning small random initial values
to weights and presenting training inputs. The connection weights that produce
strong responses are increased according to

$$a_\lambda^\gamma(k,\kappa,i,j) = a_\lambda^{\gamma-1}(k,\kappa,i,j) + \Delta a_\lambda^\gamma(k,\kappa,i,j)$$
$$\Delta a_\lambda^\gamma(k,\kappa,i,j) = q_\lambda \cdot f_{\lambda-1}(i,j) \cdot c_{\lambda-1}^\gamma(\kappa, m_\gamma + i - 1, n_\gamma + j - 1) \geq 0$$

$$b_\lambda^\gamma(k) = b_\lambda^{\gamma-1}(k) + \Delta b_\lambda^\gamma(k)$$
$$\Delta b_\lambda^\gamma(k) = q_\lambda \cdot v_\lambda^\gamma(m_\gamma, n_\gamma) \geq 0$$

where $\gamma$ is an update index. Restricted to a fairly small range of numbers, weights
could not be increased to the point where the contribution of the cell planes whose
initial random weights were unchanged became negligible. Those initial weights
that were not updated contribute random features to the recognition process; the
effect is that of adding noise.

## 4.2   WEIGHT NORMALIZATION

In order to reduce the effects of clipping on the quantized weights, the weight update
algorithm was modified. As can be seen from the weight update equations, the stan-
dard training procedure allows the $a_\lambda(k,\kappa,i,j)$ values to grow without bound. The
inner product of the weights and the input is normalized implicitly when computing
the s-cell output. Rather than using the available numerical range so lavishly, the
algorithm was modified to normalize the $a_\lambda(k,\kappa,i,j)$ templates explicitly during
training after they reached a prespecified bound. The reduction in classification
performance as computational precision decreases is compared between neocogni-
trons trained using the modified algorithm and networks trained using the original
algorithm in Figure 4(b). Networks trained using the modified algorithm have

somewhat higher (less than 5 percent) rejection and error rates compared to original networks when using floating point arithmetic, but demonstrate significantly better performance when computational precision is limited to eight bits or less.

## 5  SUMMARY

We have presented a CCD architecture that is well matched to the computational requirements of shared-weight neural networks with local connectivity. The implementation of the neocognitron, a shared-weight network for pattern recognition and feature extraction, was simulated and a new training procedure that significantly improves classification when limited precision arithmetic is used, is presented.

### Acknowledgements

This work was supported by the Office of Naval Resarch, DARPA, and the Department of the Air Force.

### References

A. M. Chiang and J. R. LaFranchise, "A Programmable Image Processor," to appear in the *ISSCC Digest of Technical Papers 1991*.

M. L. Chuang, *A Study of the Neocognitron Pattern Recognition Algorithm*. Master's Thesis, Massachusetts Institute of Technology, Dept. of Electrical Engineering and Computer Science, Cambridge, MA, June 1990.

K. Fukushima, "A Neural Network for Visual Pattern Recogniton," *IEEE Computer*, vol. 21, no. 3. pp. 65-75, March, 1988.

Y. Le Cun, "Generalization and Network Design Strategies," *Technical Report CRG-TR-89-4*, Department of Computer Science, University of Toronto, 1989.

Y. Le Cun, B. Boser, J. Denker, J. Henderson, D. Howard, R. Hubbard, and L. Jackel, "Handwritten Digit Recognition with a Back-Propagation Network," in D. S. Touretzky (ed.), *Advances in Neural Information Processing Systems 2*, pp. 396-404, San Mateo, CA: Morgan Kaufmann, 1989.

G. L. Martin and J. A. Pittman, "Recognizing Hand-Printed Letters and Digits," in D. S. Touretzky (ed.), *Advances in Neural Information Processing Systems 2*, pp. 405-414, San Mateo, CA: Morgan Kaufmann, 1989.

A. Waibel, T. Hanazawa, G. Hinton, K. Shikano, and K. J. Lang, "Phoneme Recognition Using Time-Delay Neural Networks," *IEEE Trans. on Acoustics, Speech and Signal Processing*, vol. 37, no. 3, pp. 329-339, March 1989.